# Gaussianization

**Scott Shaobing Chen**
Renaissance Technologies
East Setauket, NY 11733
*schen@rentec.com*

**Ramesh A. Gopinath**
IBM T.J. Watson Research Center
Yorktown Heights, NY 10598
*rameshg@us.ibm.com*

## Abstract

High dimensional data modeling is difficult mainly because the so-called "curse of dimensionality". We propose a technique called "Gaussianization" for high dimensional density estimation, which alleviates the curse of dimensionality by exploiting the independence structures in the data. Gaussianization is motivated from recent developments in the statistics literature: projection pursuit, independent component analysis and Gaussian mixture models with semi-tied covariances. We propose an iterative Gaussianization procedure which converges weakly: at each iteration, the data is first transformed to the least dependent coordinates and then each coordinate is marginally Gaussianized by univariate techniques. Gaussianization offers density estimation sharper than traditional kernel methods and radial basis function methods. Gaussianization can be viewed as efficient solution of nonlinear independent component analysis and high dimensional projection pursuit.

## 1 Introduction

*Density Estimation* is a fundamental problem in statistics. In the statistics literature, the univariate problem is well-understood and well-studied. Techniques such as (variable) kernel methods, radial basis function methods, Gaussian mixture models, etc, can be applied successfully to obtain univariate density estimates. However, the high dimensional problem is very challenging, mainly due to the problem of the so-called "curse of dimensionality". In high dimensional space, data samples are often sparsely distributed: it requires very large neighborhoods to achieve sufficient counts, or the number of samples has to grows exponentially according to the dimensions in order to achieve sufficient coverage of the sampling space. As a result, direct extension of univariate techniques can be highly biased, because they are neighborhood-based.

In this paper, we attempt to overcome the curse of dimensionality by exploiting independence structures in the data. We advocate the notion that

*Independence lifts the curse of dimensionality!*

Indeed, if the dimensions are independent, then there is no curse of dimensionality since the high dimensional problem can be reduced to univariate problems along each dimension.

For natural data sets which do not have independent dimensions, we would like to construct transforms such that after the transformation, the dimensions become independent. We propose a technique called "Gaussianization" which finds and exploits independence structures

in the data for high dimensional density estimation. For a random variable $\mathbf{X} \in \mathcal{R}^D$, we define its Gaussianization transform to be an invertible and differential transform $T(\mathbf{X})$ such that the transformed variable $T(\mathbf{X})$ follows the standard Gaussian distribution:

$$T(\mathbf{X}) \sim N(\mathbf{0}, \mathbf{I})$$

It is clear that density estimates can be derived from Gaussianization transforms. We propose an iterative procedure which converges weakly in probability: at each iteration, the data is first transformed to the *least dependent* coordinates and then each coordinate is marginally Gaussianized by univariate techniques which are based on univariate density estimation. At each iteration, the coordinates become less dependent in terms of the mutual information, and the transformed data samples become more Gaussian in terms of the Kullback-Leibler divergence. In fact, at each iteration, as far as the data is linearly transformed to less dependent coordinates, the convergence result still holds. Our convergence proof of Gaussianization is highly related to Huber's convergence proof of projection pursuit [4].

Algorithmically, each Gaussianization iteration amounts to performing the linear independent component analysis. Since the assumption of linear independent component analysis may not be valid, the resulting linear transform does not necessarily make the coordinate independent, however, it does make the coordinates as independent as possible. Therefore the *engine* of our algorithm is the linear independent component analysis. We propose an efficient EM algorithm which jointly estimates the linear transform and the marginal univariate Gaussianization transform at each iteration. Our parametrization is identical to the independence factor analysis proposed by Attias (1999) [1]. However, we apply the variational method in the M-step, as in the semi-tied covariance algorithm proposed for Gaussian mixture models by Gales (1999) [3].

## 2 Existence of Gaussianization

We first show the existence of Gaussianization transforms. Denote $\phi(\cdot)$ the probability density function of the standard normal $N(\mathbf{0}, \mathbf{I})$; denote $\phi(\cdot, \mu, \boldsymbol{\Sigma})$ the probability density function of $N(\mu, \boldsymbol{\Sigma})$; denote $\Phi(\cdot)$ the cumulative distribution function (CDF) of the standard normal.

### 2.1 Univariate Gaussianization

Univariate Gaussianization exists uniquely and can be derived from univariate density estimation. Let $X \in \mathcal{R}^1$ be the univariate variable. We assume that the density function of $X$ is strictly positive and differentiable. Let $F(\cdot)$ be the cumulative distribution function of $X$. Then $T(\cdot)$ is a Gaussianization transform if and only if it satisfies the following partial differential equation:

$$p(\mathbf{x}) = \phi(T(\mathbf{x}))|\frac{\partial T}{\partial \mathbf{x}}| \, .$$

It can be easily verified that the above partial differential equation has only two solutions:

$$\pm \Phi^{-1}(F(X)) \sim N(0, 1) \tag{1}$$

In practice, the CDF $F(\cdot)$ is not available; it has to be estimated from the training data. We choose to approximate it by Gaussian mixture models: $p(x) = \sum_{i=1}^{I} \pi_i \phi(x, \mu_i, \sigma_i^2)$; equivalently we assume the CDF $F(x) = \sum_{i=1}^{I} \pi_i \Phi(\frac{x-\mu_i}{\sigma_i})$ where the parameters $\{\pi_i, \mu_i, \sigma_i\}$ can be estimated via maximum likelihood using the standard EM algorithm. Therefore we can parameterize the Gaussianization transform as

$$T(X) = \Phi^{-1}(\sum_{i=1}^{I} \pi_i \Phi(\frac{X - \mu_i}{\sigma_i})) \tag{2}$$

In practice there is an issue of model selection: we suggest to use model selection techniques such as the Bayesian information criterion [6] to determine the number of Gaussians $I$. Throughout the paper, we shall assume that univariate density estimation and univariate Gaussianization can be solved by univariate Gaussian mixture models.

## 2.2 High Dimensional Gaussianization

However, the existence of high dimensional Gaussianization is non-trivial. We present here a theoretical construction. For simplicity, we consider the two dimensional case. Let $\mathbf{X} = (X_1, X_2)^T$ be the random variable. Gaussianization can be achieved in two steps. We first marginally Gaussianize the first coordinate $X_1$ and fix the second coordinate $X_2$ unchanged; the transformed variable will have the following density

$$p(x_1, x_2) = p(x_1)p(x_2|x_1) = \phi(x_1)p(x_2|x_1) \, .$$

We then marginally Gaussian each conditional density $p(\cdot|x_1)$ for each $x_1$. Notice that the marginal Gaussianization is different for different $x_1$:

$$T_{x_1}(X_2) = \Phi^{-1}(F_{\cdot|x_1}(X_2)) \, .$$

Once all the conditional densities are marginally Gaussianized, we achieve joint Gaussianization

$$p(x_1, x_2) = p(x_1)p(x_2|x_1) = \phi(x_1)\phi(x_2) \, .$$

The existence of high dimensional Gaussianization can be proved by similar construction.

The above construction, however, is not practical since the marginal Gaussianization of the conditional densities $p(X_2 = x_2|X_1 = x_1)$ requires estimation of the conditional densities given all $x_1$, which is impossible with finite samples. In the following sections, we shall develop an iterative Gaussianization algorithm that is practical and also can be proved to converge weakly.

High-dimensional Gaussianization is unique up to any invertible transforms which preserve the measure on $\mathcal{R}^D$ induced by the standard Gaussian distribution. Examples of such transforms are orthogonal linear transforms and certain nontrivial Nonlinear transforms.

## 3 Gaussianization with Linear ICA Assumption

Let $(\mathbf{x}_1, \cdots, \mathbf{x}_N)$ be the i.i.d. samples from the random variable $\mathbf{X} \in \mathcal{R}^D$. We assume that there exist a linear transform $A_{D \times D}$ such that the transformed variable $\mathbf{Y} = (Y_1, \cdots, Y_D)^T = A\mathbf{X}$ has independent components: $p(y_1, \cdots, y_D) = p(y_1) \cdots p(y_D)$. In this case, Gaussianization is reduced to linear ICA: we can first find the linear transformation $A$ by linear independent component analysis, and then Gaussianize each individual dimension of $\mathbf{Y}$ by univariate Gaussianization.

We parametrize the marginal Gaussianization by univariate Gaussian mixtures (2). This amounts to model the coordinates of the transformed variable by univariate Gaussian mixtures: $p(y_d) = \sum_{i=1}^{I_d} \pi_{d,i}\phi(y_d, \mu_{d,i}, \sigma_{d,i}^2)$. We would like to jointly optimize both the linear transform $A$ and the marginal Gaussianization parameters $(\pi, \mu, \sigma)$ via maximum likelihood. In fact, this is the same parametrization as in Attias (1999) [1]. We point out that modeling the coordinates after the linear transform as non-Gaussian distributions, for which we assume univariate Gaussian mixtures are adequate, leads to ICA while as modeling them as single Gaussians leads to PCA.

The joint estimation of the parameters can be computed via the EM algorithm. The auxiliary function which has to be maximized in the M-step has the following form:

$$Q(A, \pi, \mu, \sigma) = N \log |det(A)| + \sum_{n=1}^{N} \sum_{d=1}^{D} \sum_{i=1}^{I_d} w_{n,d,i} [\log \pi_{d,i} - \frac{1}{2} \log 2\pi\sigma_{d,i}^2 - \frac{(y_{n,d} - \mu_{d,i})^2}{2\sigma_{d,i}^2}]$$

where $(w_{n,d,i})$ are the posterior counts computed at the E-step. It can be easily shown that the priors $(\pi_{d,i}$ can be easily updated and the means $(\mu_{d,i}$ can be entirely determined by the linear transform $A$. However, updating the linear transform $A$ and the variances $(\sigma_{d,i})$ does not have closed form solution and has to be solved iteratively by numerical methods.

Attias (1999) [1] proposed to optimize $Q$ via gradient descent: at each iteration, one fixes the linear transform and compute the Gaussian mixture parameters, then fixes the Gaussian mixture parameters and update the linear transform via gradient descent using the so-called natural gradient.

We propose an iterative algorithm as in Gales (1999) [3] for the M-step which does not involve gradient descent and the nuisance and instability caused by of the step size parameter. At each iteration, we fix the linear transform $A$ and update the variances $(\sigma_{d,i})$; we then fix $(\sigma_{d,i})$ and update each row of $A$ with all the other rows of $A$ fixed: updating each row amounts to solving a system of linear equations. Our iterative scheme guarantees that the auxiliary function $Q$ to be increased at every iteration. Notice that each iteration in our M-step updates the rows of the linear matrix $A$ by solving $D$ linear equations. Although our iterative scheme may be slightly more expensive per iteration than standard numerical optimization techniques such as Attias' algorithm, in practice it converges after very few iterations, as observed in Gales (1999) [3]. In contrast the numerical optimization scheme may take an order of magnitude more iterations. In fact, in our experiments, our algorithm converges much faster than Attias's algorithm. Furthermore, our algorithm is stable since each iteration is guaranteed to increase the likelihood.

The M-step in both Attias' algorithm and our algorithm can be implemented efficiently by storing and accessing the sufficient statistics. Typically in our M-steps, most of the improvement on the likelihood comes in the first few iterations. Therefore we can stop each M-step after, say one iteration of updating the parameters; even though the auxiliary function is not optimized, but it is guaranteed to improve. Therefore we obtained the so-called generalized EM algorithm. Attias (1999) [1] reported faster convergence of the generalized EM algorithm than the standard EM algorithm.

## 4  Iterative Gaussianization

In this section we develop an iterative algorithm which Gaussianizes arbitrary random variables. At each iteration, the data is first transformed to the *least dependent* coordinates and then each coordinate is marginally Gaussianized by univariate techniques which are based on univariate density estimation. We shall show that transforming the data into the least dependent coordinates can be achieved by linear independent component analysis. We also prove the weak convergence result.

We define the negentropy [1] of a random variable $\mathbf{X} = (X_1, \cdots, X_D)^T$ as the Kullback-Leibler divergence between $\mathbf{X}$ and the standard Gaussian distribution. We define the marginal negentropy to be $J_M(\mathbf{X}) = \sum_{d=1}^{D} J(X_d)$. One can show that the negentropy can be decomposed as the sum of the marginal negentropy and the mutual information: $J(\mathbf{X}) = J_M(\mathbf{X}) + I(\mathbf{X})$. Gaussianization is equivalent to finding an invertible transform $T(\cdot)$ such that the negentropy of the transformed variable vanishes: $J(T(\mathbf{X})) = 0$.

For arbitrary random variable $\mathbf{X} \in \mathcal{R}^D$, we propose the following iterative Gaussianization algorithm. Let $\mathbf{X}^{(0)} = \mathbf{X}$. At each iteration,

(A)  Linearly transform the data: $\mathbf{Y}^{(k)} = A\mathbf{X}^{(k)}$.

(B) Nonlinearly transform the data by marginal Gaussianization:

$$\mathbf{X}^{(k+1)} = \Psi_{\pi,\mu,\sigma}(\mathbf{Y}^{(k)})$$

where the marginal Gaussianization $\Psi_{\pi,\mu,\sigma}(\cdot)$, which approximates the ideal marginal Gaussianization $\Psi(\cdot)$, can be derived from univariate Gaussian mixtures (2):

$$X_d^{(k+1)} = \Phi^{-1}\left(\sum_{i=1}^{I_d} \pi_{d,i}\Phi\left(\frac{\mathbf{Y}_d^{(k)} - \mu_{d,i}}{\sigma_{d,i}}\right)\right)$$

The parameters are chosen by minimizing the negentropy of the transformed variable $\mathbf{X}^{(k+1)}$:

$$(\hat{A}, \hat{\pi}, \hat{\mu}, \hat{\sigma}) = \min_{A,\pi,\mu,\sigma} J(\Psi_{\pi,\mu,\sigma}(A\mathbf{X})) . \tag{3}$$

Thus, after each iteration, the transformed variable becomes as close as possible to the standard Gaussian in the Kullback-Leibler distance.

First, the problem of minizing the negentropy (3) is equivalent to the maximum likelihood problem for Gaussianization with linear ICA assumption in section 3, and thus can be solved by the same efficient EM algorithm.

Second, since the data $\mathbf{X}^{(k)}$ might not satisfy the linear ICA assumption, the optimal linear transform might not transform $\mathbf{X}^{(k)}$ into independent coordinates. However, it does transform $\mathbf{X}^{(k)}$ into the *least dependent* coordinates, since

$$J(\mathbf{X}^{(k+1)}) = J_M(\Psi(A\mathbf{X}^{(k)})) + I(\Psi(A\mathbf{X}^{(k)})) = I(A\mathbf{X}^{(k)}) .$$

Further more, if the linear transform $A$ is constrained to be orthogonal, then finding the least dependent coordinates is equivalent to finding the marginally most non-Gaussian coordinates, since

$$J(\mathbf{X}^{(k)}) = J(A\mathbf{X}^{(k)}) = J_M(A\mathbf{X}^{(k)}) + I(A\mathbf{X}^{(k)})$$

(notice that the negentropy is invariant under orthogonal transforms).

Therefore our iterative algorithm can be viewed as follows. At each iteration, the data is linearly transformed to the least dependent coordinates and then each coordinate is marginally Gaussianized. In practice, after the first iteration, the algorithm finds linear transforms which are almost orthogonal. Therefore one can also view practically that at each iteration, the data is linearly transformed to the most marginally non-Gaussian coordinates and then each coordinate is marginally Gaussianized.

For the sake of simplicity, we assume that we can achieve perfect marginal Gaussianization $\Psi(\cdot)$ by $\Psi_{\pi,\mu,\sigma}(\cdot))$, which is derived from univariate Gaussian mixtures. In fact, when the number of Gaussians goes to infinity and the number of samples goes to infinity, one can show that

$$\lim \Psi_{\pi,\mu,\sigma} = \Psi.$$

Thus it suffices to analyze the ideal iterative Gaussianization

$$\mathbf{X}^{(k)} = \Psi(A\mathbf{X}^{(k)})$$

where

$$A = argmin\ J(\Psi(A\mathbf{X}^{(k)})) = argmin\ I(A\mathbf{X}^{(k)}) .$$

Following Huber's argument [4], we can show that

$$\mathbf{X}^{(k)} \to N(\mathbf{0}, \mathbf{I})$$

in the sense of weak convergence, i.e. the density function of $\mathbf{X}^{(k)}$ converges pointwise to the density function of standard normal.

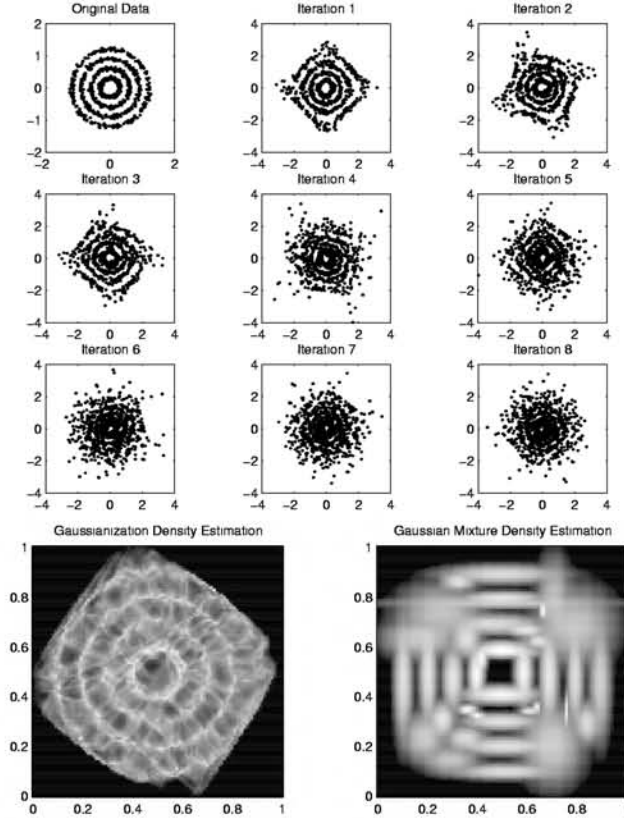

Figure 1: Iterative Gaussianization on a synthetic circular data set

We point out that out iterative algorithm can be relaxed as follows. At each iteration, the data can linearly transformed into coordinates which are *less* dependent, instead of into coordinates which are the *least* dependent:

$$I(\mathbf{X}^{(k)} - I(A_k\mathbf{X}^{(k)}) \geq \epsilon[I(\mathbf{X}^{(k)} - \inf_A I(A\mathbf{X}^{(k)})]$$

where the constant $\epsilon > 0$. We can show that this relaxed algorithm still converges weakly.

## 5  Examples

We demonstrate the process of our iterative Gaussianization algorithm through a very difficult two dimensional synthetic data set. The true underlying variable is circularly distributed: in the polar coordinate system, the angle is uniformly distributed; the radius follows a mixture of four non-overlapping Gaussians. We drew 1000 i.i.d. samples from this distribution. We ran 8 iterations to Gaussianize the data set. Figure 4 displays the transformed data set at each iteration. Clearly we see the transformed data gradually becomes standard Gaussian.

Let $\mathbf{X}^{(0)} = \mathbf{X}$; assume that the iterative Gaussianization procedure converges after $K$ iterations, i.e. $\mathbf{X}^{(K)} \sim N(\mathbf{0}, \mathbf{I})$. Since the transforms at each iteration are invertible, we can then compute Jacobian and obtain density estimation for $\mathbf{X}$. The Jacobian can be computed rapidly due to the chain rule. Figure 4 compares the Gaussianization density

estimate (8 iterations) and Gaussian mixture density estimate (40 Gaussians). Clearly we see that the Gaussianization density estimate recovers the four circular structure; however, the Gaussian mixture estimate lacks resolution.

## 6   Discussion

Gaussianization is closely connected with the exploratory projection pursuit algorithm proposed by Friedman (1987) [2]. In fact we argue that our iterative Gaussianization procedure can easily constrained as an efficient parametric solution of high dimensional projection pursuit. Assume that we are interested in $l$-dimensional projections where $1 \leq l \leq D$. If we constrain that at each iteration the linear transform has to be orthogonal, and only the first $l$ coordinates of the transformed variable are marginally Gaussianized, then the iterative Gaussianization algorithm achieves $l$ dimensional projection pursuit. The bottleneck of Friedman's high dimensional projection pursuit is to find the *jointly* most non-Gaussian projection and to *jointly* Gaussianize that projection. In contrast, our algorithm finds the most *marginally* non-Gaussian projection and *marginally* Gaussianize that projection; it can be computed by an efficient EM algorithm.

We argue that Gaussianization density estimation indeed alleviates the problem of the curse of dimensionality. At each iteration, the effect of the curse of dimensionality is solely on finding a linear transform such that the transformed coordinates are less dependent, which is a relatively much easier problem than the original problem of high dimensional density estimation itself; after the linear transform, the marginal Gaussianization can be derived from univariate density estimation, which has nothing to do with the curse of dimensionality. Hwang 1994 [5] performed extensive comparative study among the following three popular density estimates: one dim projection pursuit density estimates (a special case of our iterative Gaussianization algorithm), adaptive kernel density estimates and radial basis function density estimates; he concluded that projection pursuit density estimates outperform in most data set.

We are currently experimenting with application of Gaussianization density estimation in automatic speech and speaker recognition.

## Footnotes

[1] We are abusing the terminology slightly: normally the negentropy of a random variable is defined to be the Kullback-Leibler distance between itself and the Gaussian variable with the same mean and covariance.

## References

[1] H. Attias, "Independent factor analysis", *Neural Computation*, vol. 11, pp. 803-851, 1999.

[2] J.H. Friedman, "Exploratory projection pursuit", *J. American Statistical Association*, vol. 82, pp. 249-266, 1987.

[3] M.J.F. Gales, "Semi-tied covariance matrices for hidden Markov Models", *IEEE Transaction Speech and Audio Processing*, vol. 7, pp. 272-281, 1999.

[4] P.J. Huber, "Projection pursuit", *Annals of Statistics*, vol. 13, pp 435-525, 1985.

[5] J. Hwang, S. Lay and A. Lippman, "Nonparametric multivariate density estimation: a comparative study", *IEEE Transaction Signal Processing*, vol. 42, pp 2795-2810, 1994.

[6] G. Schwarz, "Estimating the dimension of a model", *Annals of Statistics*, vol. 6, pp 461-464, 1978.
